# An Information-theoretic Learning Algorithm for Neural Network Classification

**David J. Miller**
Department of Electrical Engineering
The Pennsylvania State University
State College, Pa. 16802

**Ajit Rao, Kenneth Rose, and Allen Gersho**
Department of Electrical and Computer Engineering
University of California
Santa Barbara, Ca. 93106

## Abstract

A new learning algorithm is developed for the design of statistical classifiers minimizing the rate of misclassification. The method, which is based on ideas from information theory and analogies to statistical physics, assigns data to classes *in probability*. The distributions are chosen to minimize the expected classification error while simultaneously enforcing the classifier's structure and a level of "randomness" measured by Shannon's entropy. Achievement of the classifier structure is *quantified* by an associated cost. The constrained optimization problem is equivalent to the minimization of a Helmholtz free energy, and the resulting optimization method is a basic extension of the deterministic annealing algorithm that explicitly enforces structural constraints on assignments while reducing the entropy and expected cost with temperature. In the limit of low temperature, the error rate is minimized directly and a hard classifier with the requisite structure is obtained. This learning algorithm can be used to design a variety of classifier structures. The approach is compared with standard methods for radial basis function design and is demonstrated to substantially outperform other design methods on several benchmark examples, while often retaining design complexity comparable to, or only moderately greater than that of strict descent-based methods.

# 1 Introduction

The problem of designing a statistical classifier to minimize the probability of misclassification or a more general risk measure has been a topic of continuing interest since the 1950s. Recently, with the increase in power of serial and parallel computing resources, a number of complex neural network classifier structures have been proposed, along with associated learning algorithms to design them. While these structures offer great potential for classification, this potential cannot be fully realized without effective learning procedures well-matched to the minimum classification-error objective. Methods such as back propagation which approximate class targets in a squared error sense do not directly minimize the probability of error. Rather, it has been shown that these approaches design networks to approximate the class a posteriori probabilities. The probability estimates can then be used to form a decision rule. While large networks can in principle accurately approximate the Bayes discriminant, in practice the network size must be constrained to avoid overfitting the (finite) training set. Thus, discriminative learning techniques, e.g. (Juang and Katagiri, 1992), which seek to directly minimize classification error may achieve better results. However, these methods may still be susceptible to finding shallow local minima far from the global minimum.

As an alternative to strict descent-based procedures, we propose a new deterministic learning algorithm for statistical classifier design with a demonstrated potential for avoiding local optima of the cost. Several deterministic, annealing-based techniques have been proposed for avoiding nonglobal optima in computer vision and image processing (Yuille, 1990), (Geiger and Girosi,1991), in combinatorial optimization, and elsewhere. Our approach is derived based on ideas from information theory and statistical physics, and builds on the probabilistic framework of the deterministic annealing (DA) approach to clustering and related problems (Rose et al., 1990,1992,1993). In the DA approach for data clustering, the probability distributions are chosen to minimize the expected clustering cost, given a constraint on the level of randomness, as measured by Shannon's entropy [1].

In this work, the DA approach is extended in a novel way, most significantly *to incorporate structural constraints on data assignments*, but also to minimize the probability of error as the cost. While the general approach we suggest is likely applicable to problems of structured vector quantization and regression as well, we focus on the classification problem here. Most design methods have been developed for specific classifier structures. In this work, we will develop a general approach but only demonstrate results for RBF classifiers. The design of nearest prototype and MLP classifiers is considered in (Miller et al., 1995a,b). Our method provides substantial performance gains over conventional designs for all of these structures, while retaining design complexity in many cases comparable to the strict descent methods. Our approach often designs small networks to achieve training set performance that can only be obtained by a much larger network designed in a conventional way. The design of smaller networks may translate to superior performance outside the training set.

## 2   Classifier Design Formulation

### 2.1   Problem Statement

Let $\mathcal{T} = \{(\mathbf{x}, c)\}$ be a training set of $N$ labelled vectors, where $\mathbf{x} \in \mathcal{R}^n$ is a feature vector and $c \in \mathcal{I}$ is its class label from an index set $\mathcal{I}$. A classifier is a mapping $C : \mathcal{R}^n \to \mathcal{I}$, which assigns a class label in $\mathcal{I}$ to each vector in $\mathcal{R}^n$. Typically, the classifier is represented by a set of model parameters $\Lambda$. The classifier specifies a partitioning of the feature space into regions $R_j \equiv \{\mathbf{x} \in \mathcal{R}^n : C(\mathbf{x}) = j\}$, where $\bigcup_j R_j \equiv \mathcal{R}^n$ and $\bigcap_j R_j \equiv \emptyset$. It also induces a partitioning of the training set into sets $\mathcal{T}_j \subset \mathcal{T}$, where $\mathcal{T}_j \equiv \{\{\mathbf{x}, c\} : \mathbf{x} \in R_j, (\mathbf{x}, c) \in \mathcal{T}\}$. A training pair $(\mathbf{x}, c) \in \mathcal{T}$ is *misclassified* if $C(\mathbf{x}) \neq c$. The performance measure of primary interest is the empirical error fraction $P_e$ of the classifier, i.e. the fraction of the training set (for generalization purposes, the fraction of the test set) which is misclassified:

$$P_e = \frac{1}{N} \sum_{(\mathbf{x}, c) \in \mathcal{T}} \delta(c, C(\mathbf{x})) = \frac{1}{N} \sum_{j \in \mathcal{I}} \sum_{(\mathbf{x}, c) \in \mathcal{T}_j} \delta(c, j), \tag{1}$$

where $\delta(c, j) = 1$ if $c \neq j$ and 0 otherwise. In this work, we will assume that the classifier produces an output $F_j(\mathbf{x})$ associated with each class, and uses a "winner-take-all" classification rule:

$$R_j \equiv \{\mathbf{x} \in \mathcal{R}^n : F_j(\mathbf{x}) \geq F_k(\mathbf{x}) \;\; \forall k \in \mathcal{I}\}. \tag{2}$$

This rule is consistent with MLP and RBF-based classification.

### 2.2   Randomized Classifier Partition

As in the original DA approach for clustering (Rose et al., 1990,1992), we cast the optimization problem in a framework in which data are assigned to classes *in probability*. Accordingly, we define the *probabilities of association* between a feature $\mathbf{x}$ and the class regions, *i.e.* $\{P[\mathbf{x} \in R_j]\}$. As our design method, which optimizes over these probabilities, must ultimately form a classifier that makes "hard" decisions based on a specified network model, the distributions must be chosen to be consistent with the decision rule of the model. In other words, we need to introduce randomness into the classifier's partition. Clearly, there are many ways one could define probability distributions which are consistent with the hard partition at some limit. We use an information-theoretic approach. We measure the randomness or uncertainty by Shannon's entropy, and determine the distribution for a given level of entropy. At the limit of zero entropy we should recover a hard partition. For now, suppose that the values of the model parameters $\Lambda$ have been fixed. We can then write an objective function whose maximization determines the hard partition for a given $\Lambda$:

$$F_h = \frac{1}{N} \sum_{j \in \mathcal{I}} \sum_{(\mathbf{x}, c) \in \mathcal{T}_j} F_j(\mathbf{x}). \tag{3}$$

Note specifically that maximizing (3) over all possible partitions captures the decision rule of (2). The probabilistic generalization of (3) is

$$F = \frac{1}{N} \sum_{(\mathbf{x}, c) \in \mathcal{T}} \sum_j P[\mathbf{x} \in R_j] F_j(\mathbf{x}), \tag{4}$$

where the (randomized) partition is now represented by association probabilities, and the corresponding entropy is

$$H = -\frac{1}{N} \sum_{(\mathbf{x}, c) \in \mathcal{T}} \sum_j P[\mathbf{x} \in R_j] \log P[\mathbf{x} \in R_j]. \tag{5}$$

We determine the distribution at a given level of randomness as the one which maximizes $F$ while maintaining $H$ at a prescribed level $\hat{H}$:

$$\max_{\{P[\mathbf{X} \in R_j]\}} F \quad \text{subject to} \quad H = \hat{H}. \tag{6}$$

The result is the *best* probabilistic partition, in the sense of $F$, *at the specified level of randomness*. For $\hat{H} = 0$ we get back the hard partition maximizing (3). At any $\hat{H}$, the solution of (6) is the Gibbs distribution

$$P[\mathbf{x} \in R_j] \equiv P_{j|x}(\Lambda) = \frac{e^{\gamma F_j(\mathbf{X})}}{\sum_k e^{\gamma F_k(\mathbf{X})}}, \tag{7}$$

where $\gamma$ is the Lagrange multiplier. For $\gamma \to 0$, the associations become increasingly uniform, while for $\gamma \to \infty$, they revert to hard classifications, equivalent to application of the rule in (2). Note that the probabilities depend on $\Lambda$ through the network outputs. Here we have emphasized this dependence through our choice of concise notation.

## 2.3  Information-Theoretic Classifier Design

Until now we have formulated a controlled way of introducing randomness into the classifier's partition while enforcing its structural constraint. However, the derivation assumed that the model parameters were given, and thus produced only the *form* of the distribution $P_{j|x}(\Lambda)$, without actually prescribing how to choose the values of its parameter set. Moreover the derivation did not consider the ultimate goal of minimizing the probability of error. Here we remedy both shortcomings.

The method we suggest gradually enforces formation of a hard classifier minimizing the probability of error. We start with a highly random classifier and a high expected misclassification cost. We then gradually reduce both the randomness and the cost in a deterministic learning process which enforces formation of a hard classifier with the requisite structure. As before, we need to introduce randomness into the partition while enforcing the classifier's structure, only now we are also interested in minimizing the expected misclassification cost. While satisfying these multiple objectives may appear to be a formidable task, the problem is greatly simplified by restricting the choice of random classifiers to the set of distributions $\{P_{j|x}(\Lambda)\}$ as given in (7) – these random classifiers naturally enforce the structural constraint through $\gamma$. Thus, from the parametrized set $\{P_{j|x}(\Lambda)\}$, we seek that distribution which minimizes the average misclassification cost while constraining the entropy:

$$\min_{\gamma, \Lambda} < P_e > \equiv \min_{\gamma, \Lambda} \frac{1}{N} \sum_{(\mathbf{X}, c) \in \mathcal{T}} \sum_j P_{j|x}(\Lambda) \delta(c, j), \tag{8}$$

subject to

$$H = \hat{H}.$$

The solution yields the *best* random classifier in the sense of minimum $< P_e >$ for a given $\hat{H}$. At the limit of zero entropy, we should get the best *hard* classifier in the sense of $P_e$ with the desired structure, i.e. satisfying (2).

The constrained minimization (8) is equivalent to the unconstrained minimization of the Lagrangian:

$$\min_{\Lambda, \gamma} L \equiv \min_{\Lambda, \gamma} \beta < P_e > -H, \tag{9}$$

where $\beta$ is the Lagrange multiplier associated with (8). For $\beta = 0$, the sole objective is entropy maximization, which is achieved by the uniform distribution. This solution, which is the global minimum for $L$ at $\beta = 0$, can be obtained by choosing $\gamma = 0$. At the other end of the spectrum, for $\beta \to \infty$, the sole objective is to minimize $< P_e >$, and is achieved by choosing a non-random (hard) classifier (hence minimizing $P_e$). The hard solution satisfies the classification rule (2) and is obtained for $\gamma \to \infty$.

Motivation for minimizing the Lagrangian can be obtained from a physical perspective by noting that $L$ is the Helmholtz free energy of a simulated system, with $< P_e >$ the "energy", $H$ the system entropy, and $\frac{1}{\beta}$ the "temperature". Thus, from this physical view we can suggest a deterministic annealing (DA) process which involves minimizing $L$ starting at the global minimum for $\beta = 0$ (high temperature) and tracking the solution while increasing $\beta$ towards infinity (zero temperature). In this way, we obtain a sequence of solutions of decreasing entropy and average misclassification cost. Each such solution is the best random classifier in the sense of $< P_e >$ for a given level of randomness. The annealing process is useful for avoiding local optima of the cost $< P_e >$, and minimizes $< P_e >$ directly at low temperature. While this annealing process ostensibly involves the quantities $H$ and $< P_e >$, the restriction to $\{P_{j|x}(\Lambda)\}$ from (7) ensures that the process also enforces the structural constraint on the classifier in a controlled way. *Note in particular that $\gamma$ has not lost its interpretation as a Lagrange multiplier determining $F$.* Thus, $\gamma = 0$ means that $F$ is unconstrained – we are free to choose the uniform distribution. Similarly, sending $\gamma \to \infty$ requires maximizing $F$ – hence the hard solution. Since $\gamma$ is chosen to minimize $L$, this parameter effectively determines the level of $F$ – the level of structural constraint – *consistent with $H$ and $< P_e >$ for a given $\beta$*. As $\beta$ is increased, the entropy constraint is relaxed, allowing greater satisfaction of *both* the minimum $< P_e >$ and maximum $F$ objectives. Thus, annealing in $\beta$ gradually enforces both the structural constraint (via $\gamma$) and the minimum $< P_e >$ objective [2].

Our formulation clearly identifies what distinguishes the annealing approach from direct descent procedures. Note that a descent method could be obtained by simply neglecting the constraint on the entropy, instead choosing to directly minimize $< P_e >$ over the parameter set. This minimization will directly lead to a hard classifier, and is akin to the method described in (Juang and Katagiri, 1992) as well as other related approaches which attempt to directly minimize a smoothed probability of error cost. However, as we will experimentally verify through simulations, our annealing approach outperforms design based on directly minimizing $< P_e >$.

For conciseness, we will not derive necessary optimality conditions for minimizing the Lagrangian at a give temperature, nor will we specialize the formulation for individual classification structures here. The reader is referred to (Miller et al., 1995a) for these details.

## 3 Experimental Comparisons

We demonstrate the performance of our design approach in comparison with other methods for the normalized RBF structure (Moody and Darken, 1989). For the DA method, steepest descent was used to minimize $L$ at a sequence of exponentially increasing $\beta$, given by $\beta(n + 1) = \alpha\beta(n)$, for $\alpha$ between 1.05 and 1.1. We have found that much of the optimization occurs at or near a critical temperature in the

| Method | DA | | TR-RBF | | | | MD-RBF | | $\nabla P_e$ |
|---|---|---|---|---|---|---|---|---|---|
| M | 4 | 30 | 4 | 10 | 30 | 50 | 10 | 50 | 10 |
| $P_e$ (train) | 0.11 | 0.028 | 0.33 | 0.162 | 0.145 | 0.129 | 0.3 | 0.19 | 0.18 |
| $P_e$ (test) | 0.13 | 0.167 | 0.35 | 0.165 | 0.168 | 0.179 | 0.37 | 0.18 | 0.20 |

Table 1: A comparison of DA with known design techniques for RBF classification on the 40-dimensional noisy waveform data from (Breiman et al., 1980).

solution process. Beyond this critical temperature, the annealing process can often be "quenched" to zero temperature by sending $\gamma \rightarrow \infty$ without incurring significant performance loss. Quenching the process often makes the design complexity of our method comparable to that of descent-based methods such as back propagation or gradient descent on $< P_e >$.

We have compared our RBF design approach with the method in (Moody and Darken, 1989) (MD-RBF), with a method described in (Tarassenko and Roberts,1994) (TR-RBF), with the approach in (Musavi et al., 1992), and with steepest descent on $< P_e >$ (G-RBF). MD-RBF combines unsupervised learning of receptive field parameters with supervised learning of the weights from the receptive fields so as to minimize the squared distance to target class outputs. The primary advantage of this approach is its modest design complexity. However, the receptive fields are not optimized in a supervised fashion, which can cause performance degradation. TR-RBF optimizes all of the RBF parameters to approximate target class outputs. This design is more complex than MD-RBF and achieves better performance for a given model size. However, as aforementioned, the TR-RBF design objective is not equivalent to minimizing $P_e$, but rather to approximating the Bayes-optimal discriminant. While direct descent on $< P_e >$ may minimize the "right" objective, problems of local optima may be quite severe. In fact, we have found that the performance of all of these methods can be quite poor without a judicious initialization. For all of these methods, we have employed the unsupervised learning phase described in (Moody and Darken, 1989) (based on Isodata clustering and variance estimation) as model initialization. Then, steepest descent was performed on the respective cost surface. We have found that the complexity of our design is typically 1-5 times that of TR-RBF or G-RBF (though occasionally our design is actually faster than G-RBF). Accordingly, we have chosen the best results based on five random initializations for these techniques, and compared with the single DA design run.

One example reported here is the 40D "noisy" waveform data used in (Breiman et al., 1980) (obtained from the UC-Irvine machine learning database repository.). We split the 5000 vectors into equal size training and test sets. Our results in Table I demonstrate quite substantial performance gains over all the other methods, and performance quite close to the estimated Bayes rate of 14%. Note in particular that the other methods perform quite poorly for a small number of receptive fields ($M$), and need to increase $M$ to achieve training set performance comparable to our approach. However, performance on the test set does not necessarily improve, and may degrade for increasing $M$.

To further justify this claim, we compared our design with results reported in (Musavi et al., 1992), for the two and eight dimensional mixture examples. For the 2D example, our method achieved $P_{e_{train}} = 6.0\%$ for a 400 point training set and $P_{e_{test}} = 6.1\%$ on a 20,000 point test set, using $M = 3$ units (These results are near-optimal, based on the Bayes rate.). By contrast, the method of Musavi et

al. used 86 receptive fields and achieved $P_{e_{test}} = 9.26\%$. For the 8D example and $M = 5$, our method achieved $P_{e_{train}} = 8\%$ and $P_{e_{test}} = 9.4\%$ (again near-optimal), while the method in (Musavui et al., 1992) achieved $P_{e_{test}} = 12.0\%$ using $M = 128$.

In summary, we have proposed a new, information-theoretic learning algorithm for classifier design, demonstrated to outperform other design methods, and with general applicability to a variety of structures. Future work may investigate important applications, such as recognition problems for speech and images. Moreover, our extension of DA to incorporate structure is likely applicable to structured vector quantizer design and to regression modelling. These problems will be considered in future work.

## Acknowledgements

This work was supported in part by the National Science Foundation under grant no. NCR-9314335, the University of California MICRO program, DSP Group, Inc. Echo Speech Corporation, Moseley Associates, National Semiconductor Corp., Qualcomm, Inc., Rockwell International Corporation, Speech Technology Labs, and Texas Instruments, Inc.

## Footnotes

[1]Note that in (Rose et al., 1990,1992,1993), the DA method was formally derived using the maximum entropy principle. Here we emphasize the alternative, but mathematically equivalent description that the chosen distributions minimize the expected cost given constrained entropy. This formulation may have more intuitive appeal for the optimization problem at hand.

[2]While not shown here, the method does converge directly for $\beta \to \infty$, and at this limit enforces the classifier's structure.

## References

L. Breiman, J. H. Friedman, R. A. Olshen, and C. J. Stone. *Classification and Regression Trees*. The Wadsworth Statistics/Probability Series, Belmont,CA., 1980.

D. Geiger and F. Girosi. Parallel and deterministic algorithms from MRFs: Surface reconstruction. *IEEE Trans. on Patt. Anal. and Mach. Intell.*, 13:401–412, 1991.

B.-H. Juang and S. Katagiri. Discriminative learning for minimum error classification. *IEEE Trans. on Sig. Proc.*, 40:3043–3054, 1992.

D. Miller, A. Rao, K. Rose, and A. Gersho. A global optimization technique for statistical classifier design. (Submitted for publication.), 1995.

D. Miller, A. Rao, K. Rose, and A. Gersho. A maximum entropy framework for optimal statistical classification. In *IEEE Workshop on Neural Networks for Signal Processing*.), 1995.

J. Moody and C. J. Darken. Fast learning in locally-tuned processing units. *Neural Comp.*, 1:281–294, 1989.

M. T. Musavi, W. Ahmed, K. H. Chan, K. B. Faris, and D. M. Hummels. On the training of radial basis function classifiers. *Neural Networks*, 5:595–604, 1992.

K. Rose, E. Gurewitz, and G. C. Fox. Statistical mechanics and phase transitions in clustering. *Phys. Rev. Lett.*, 65:945–948, 1990.

K. Rose, E. Gurewitz, and G. C. Fox. Vector quantization by deterministic annealing. *IEEE Trans. on Inform. Theory*, 38:1249–1258, 1992.

K. Rose, E. Gurewitz, and G. C. Fox. Constrained clustering as an optimization method. *IEEE Trans. on Patt. Anal. and Mach. Intell.*, 15:785–794, 1993.

L. Tarassenko and S. Roberts. Supervised and unsupervised learning in radial basis function classifiers. *IEE Proc.-Vis. Image Sig. Proc.*, 141:210–216, 1994.

A. L. Yuille. Generalized deformable models, statistical physics, and matching problems. *Neural Comp.*, 2:1–24, 1990.